# Neuronal Regulation Implements Efficient Synaptic Pruning

**Gal Chechik and Isaac Meilijson**
School of Mathematical Sciences
Tel Aviv University, Tel Aviv 69978, Israel
ggal@math.tau.ac.il    isaco@math.tau.ac.il
**Eytan Ruppin**
Schools of Medicine and Mathematical Sciences
Tel Aviv University, Tel Aviv 69978, Israel
ruppin@math.tau.ac.il

## Abstract

Human and animal studies show that mammalian brain undergoes massive synaptic pruning during childhood, removing about half of the synapses until puberty. We have previously shown that maintaining network memory performance while synapses are deleted, requires that synapses are properly modified and pruned, removing the weaker synapses. We now show that neuronal regulation, a mechanism recently observed to maintain the average neuronal input field, results in weight-dependent synaptic modification. Under the correct range of the degradation dimension and synaptic upper bound, neuronal regulation removes the weaker synapses and judiciously modifies the remaining synapses. It implements near optimal synaptic modification, and maintains the memory performance of a network undergoing massive synaptic pruning. Thus, this paper shows that in addition to the known effects of Hebbian changes, neuronal regulation may play an important role in the self-organization of brain networks during development.

## 1 Introduction

This paper studies one of the fundamental puzzles in brain development: the massive synaptic pruning observed in mammals during childhood, removing more than half of the synapses until puberty (see [1] for review). This phenomenon is observed in various areas of the brain both in animal studies and human studies. How can the brain function after such massive synaptic elimination? what could be the computational advantage of such a seemingly wasteful developmental strategy? In

previous work [2], we have shown that synaptic overgrowth followed by judicial pruning along development improves the performance of an associative memory network with limited synaptic resources, thus suggesting a new computational explanation for synaptic pruning in childhood. The optimal pruning strategy was found to require that synapses are deleted according to their efficacy, removing the weaker synapses first.

But is there a mechanism that can implement these theoretically-derived synaptic pruning strategies in a biologically plausible manner ? To answer this question, we focus here on studying the role of neuronal regulation (NR), a mechanism operating to maintain the homeostasis of the neuron's membrane potential. NR has been recently identified experimentally by [3], who showed that neurons both up-regulate and down-regulate the efficacy of their incoming excitatory synapses in a multiplicative manner, maintaining their membrane potential around a baseline level. Independently, [4] have studied NR theoretically, showing that it can efficiently maintain the memory performance of networks undergoing synaptic degradation. Both [3] and [4] have hypothesized that NR may lead to synaptic pruning during development.

In this paper we show that this hypothesis is both computationally feasible and biologically plausible by studying the modification of synaptic values resulting from the operation of NR. Our work thus gives a possible account for the way brain networks maintain their performance while undergoing massive synaptic pruning.

## 2   The Model

*NR-driven synaptic modification (NRSM)* results from two concomitant processes: **synaptic degradation** (which is the inevitable consequence of synaptic turnover [5]), and **neuronal regulation** (NR) operating to compensate for the degradation. We therefore model NRSM by a sequence of degradation-strengthening steps. At each time step, synaptic degradation stochastically reduces the synaptic strength $W^t$ ($W^t > 0$) to $W'^{t+1}$ by

$$W'^{t+1} = W^t - (W^t)^\alpha \eta^t; \quad \eta \sim N(\mu^\eta, \sigma^\eta) \tag{1}$$

where $\eta$ is noise term with positive mean and the power $\alpha$ defines the *degradation dimension* parameter chosen in the range $[0, 1]$. Neuronal regulation is modeled by letting the post-synaptic neuron multiplicatively strengthen all its synapses by a common factor to restore its original input field

$$W^{t+1} = W'^{t+1} \frac{f_i^0}{f_i^t} \quad . \tag{2}$$

where $f_i^t$ is the input field of neuron $i$ at time $t$. The excitatory synaptic efficacies are assumed to have a viability lower bound $B^-$ below which a synapse degenerates and vanishes, and a soft upper bound $B^+$ beyond which a synapse is strongly degraded reflecting their maximal efficacy. To study of the above process in a network, a model incorporating a segregation between inhibitory and excitatory neurons (i.e. obeying Dale's law) is required. To generate this essential segregation, we modify the standard low-activity associative memory model proposed by [6] by adding a small positive term to the synaptic learning rule. In this model, $M$ memories are stored in an excitatory $N$-neuron network forming attractors of the network dynamics. The synaptic efficacy $W_{ij}$ between the $j$th (pre-synaptic) neuron and the $i$th (post-synaptic) neuron is

$$W_{ij} = \sum_{\mu=1}^{M} \left[ (\xi_i^\mu - p)(\xi_j^\mu - p) + a \right], 1 \le i \ne j \le N \quad ; \quad W_{ii} = 0 \tag{3}$$

where $\{\xi^\mu\}_{\mu=1}^M$ are $\{0,1\}$ memory patterns with coding level $p$ (fraction of firing neurons), and $a$ is some positive constant [1]. The updating rule for the state $X_i^t$ of the $i$th neuron at time $t$ is

$$X_i^{t+1} = \theta(f_i^t), \quad f_i^t = \frac{1}{N}\sum_{j=1}^N g(W_{ij})X_j^t - \frac{\mathcal{I}}{N}\sum_{j=1}^N X_j^t - T, \quad \theta(f) = \frac{1 + sign(f)}{2} \quad (4)$$

where $T$ is the neuronal threshold, and $\mathcal{I}$ is the inhibition strength. $g$ is a general modification function over the excitatory synapses, which is either derived explicitly (See Section 4), or determined implicitly by the operation of NRSM. If $g$ is linear and $\mathcal{I} = Ma$ the model reduces to the original model described by [6]. The overlap $m^\mu$ (or similarity) between the network's activity pattern $X$ and the memory $\xi^\mu$ serves to measure memory performance (retrieval acuity), and is defined as $m^\mu = \frac{1}{N}\sum_{j=1}^N(\xi_j^\mu - p)X_j$.

## 3 Neuronally Regulated Synaptic Modification

NRSM was studied by simulating the degradation-strengthening sequence in a network in which memory patterns were stored according to Eq.3. Figure 1a plots a typical distribution of synaptic values as traced along a sequence of degradation-strengthening steps (Eq. 1,2). As evident, the synaptic values diverge: some of the weights are strengthened and lie close to the upper synaptic bounds, while the other synapses degenerate and vanish. Using probabilistic considerations, it can be shown that the synaptic distribution converge to a meta-stable state where it remains for long waiting times. Figure 1b describes the metastable synaptic distribution as calculated for different $\alpha$ values.

**Evolving distribution of synaptic efficacies**
**a. Simulation results**                    **b. Numerical results**

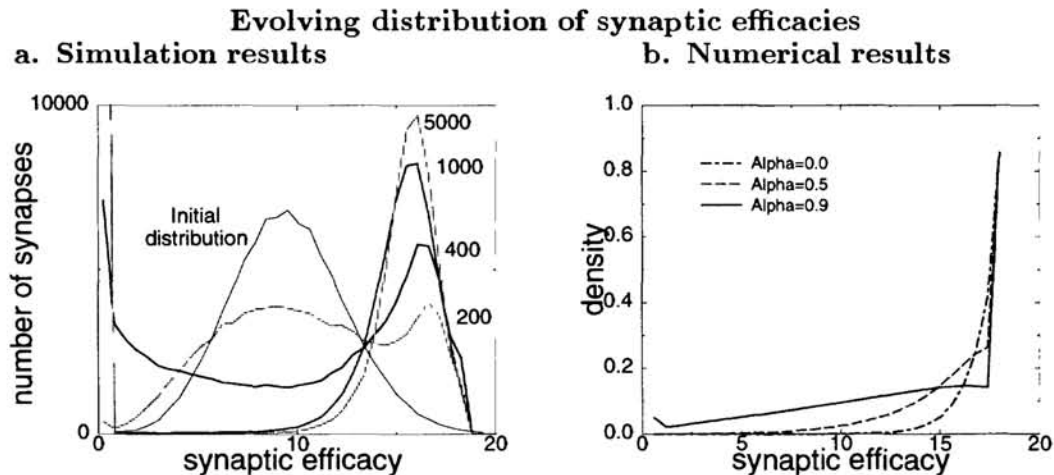

Figure 1: Distribution of synaptic strengths following a degradation-strengthening process. a) Synaptic distribution after $0, 200, 400, 1000$ and $5000$ degradation-strengthening steps of a 400 neurons network with 1000 stored memory patterns. $\alpha=0.8$, $p = 0.1$, $B^- = 10^{-5}$, $B^+ = 18$ and $\eta \sim N(0.05, 0.05)$. Qualitatively similar results were obtained for a wide range of simulation parameters. b) The synaptic distribution of the remaining synapses at the meta-stable state was calculated as the main eigen vector of the transition probability matrix.

**a. NRSM functions at the Metastable state**

**b. NRSM and random deletion**

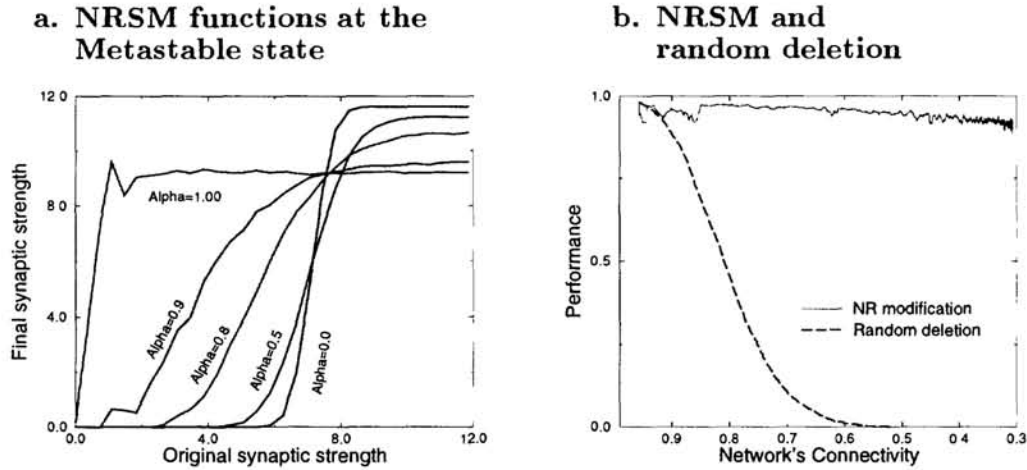

Figure 2: a) NRSM functions at the metastable state for different $\alpha$ values. Results were obtained in a 400-neurons network after performing 5000 degradation-strengthening steps. Parameter values are as in Figure 1, except $B^+ = 12$. b) Performance of NR modification and random deletion. The retrieval acuity of 200 memories stored in a network of 800 neurons is portrayed as a function of network connectivity, as the network undergoes continuous pruning until NR reaches the metastable state. $\alpha = 0$, $B^+ = 7.5$, $p = 0.1$, $m_0 = 0.80$, $a = 0.01$, $T = 0.35$, $B^- = 10^{-5}$ and $\eta \sim N(0.01, 0.01)$.

To further investigate which synapses are strengthened and which are pruned, we study the resulting synaptic modification function. Figure 2a plots the value of synaptic efficacy at the metastable state as a function of the initial synaptic efficacy, for different values of the degradation dimension $\alpha$. As observed, a sigmoidal dependency is obtained, where the slope of the sigmoid strongly depends on the degradation dimension. In the two limit cases, additive degradation ($\alpha = 0$) results in a step function at the metastable state, while multiplicative degradation ($\alpha = 1$) results in random diffusion of the synaptic weights toward a memoryless mean value. Different values of $\alpha$ and $B^+$ result in different levels of synaptic pruning: When the synaptic upper bound $B^+$ is high, the surviving synapses assume high values, leading to massive pruning to maintain the neuronal input field, which in turn reduces network's performance. Low $B^+$ values lead to high connectivity, but limit synapses to a small set of possible values, again reducing memory performance. Our simulations show that optimal memory retrieval is obtained for $B^+$ values that lead to deletion levels of $40\% - 60\%$, in which NR indeed maintains the network performance. Figure 2b traces the average retrieval acuity of a network throughout the operation of NR, versus a network subject to random deletion at the same pruning levels. While the retrieval of a randomly pruned network collapses already at low deletion levels of about 20%, a network undergoing NR performs well even in high deletion levels.

## 4   Optimal Modification In Excitatory-Inhibitory Networks

To obtain a a comparative yardstick to evaluate the efficiency of NR as a selective pruning mechanism, we derive optimal modification functions maximizing memory performance in our excitatory-inhibitory model and compare them to the NRSM functions.

We study general synaptic modification functions, which prune some of the synapses and possibly modify the rest, while satisfying global constraints on synapses such as the number or total strength of the synapses. These constraints reflect the observation that synaptic activity is strongly correlated with energy consumption in the brain [7], and synaptic resources may hence be inherently limited in the adult brain.

We evaluate the impact of these functions on the network's retrieval performance, by deriving their effect on the signal to noise ratio (S/N) of the neuron's input field (Eqs. 3,4), known to be the primary determinant of retrieval capacity ([8]). This analysis, conducted in a similar manner to [2] yields

$$\frac{S}{N} = \frac{E(f_i|\xi_i = 1) - E(f_i|\xi_i = 0)}{\sqrt{V(f_i|\xi_i)}} = \sqrt{\frac{N}{M}} \frac{m^\mu}{\sqrt{p}} \frac{E[z\widehat{g}(z)]}{\sqrt{E[\widehat{g}^2(z)] - pE^2[\widehat{g}(z)]}} \quad (5)$$

where $z \sim N(0,1)$ and $g$ is the modification function of Eq. 4 but is now explicitly applied to the synapses. To derive optimal synaptic modification functions with limited synaptic resources, we consider $g$ functions that zero all synapses except those in some set $A$, and keep the integral

$$\int_A g^k(z)\phi(z)dz \quad ; \quad k = 0, 1, \dots \quad ; \quad g(z) = 0 \, \forall z \notin A \quad ; \quad \phi(z) = \frac{e^{z^2/2}}{\sqrt{2\pi}} \quad (6)$$

limited. We then maximize the S/N under this constraint using the Lagrange method. Our results show that *without any synaptic constraints* the optimal function is the identity function, that is, the original Hebbian rule is optimal. When the *number of synapses* is restricted ($k = 0$), the optimal modification function is a linear function for all the remaining synapses

$$g(W) = aW - \mu a + b \quad \text{where} \begin{cases} a = \sqrt{\frac{1}{\int_A z^2\phi(z)dz}} \\ b = \sigma a \frac{\int_A z\phi(z)dz}{(1-\int_A \phi(z)dz)} \end{cases} \begin{cases} \mu = E(W) \\ \sigma^2 = V(W) \end{cases} \quad (7)$$

for any deletion set $A$. To find the synapses that should be deleted, we have numerically searched for a deletion set maximizing S/N while limiting $g(W)$ to positive values (as required by the segregation between excitatory and inhibitory neurons). The results show, that **weak synapses pruning**, a modification strategy that removes the weakest synapses and modifies the rest according to Eq. 7, is optimal at deletion levels above 50%. For lower deletion levels, the above $g$ function fails to satisfy the positivity constraint for any set $A$. When the positivity constraint is ignored, S/N is maximized if the weights closest to the mean are deleted and the remaining synapses are modified according to Eq 7. We name this strategy **mean synapses pruning**. Figure 3 plots the memory capacity under weak-synapses pruning (compared with random deletion and mean-synaptic pruning) showing that pruning the weak synapses performs at least near optimally for lower deletion levels as well. Even more interesting, under the correct parameter values weak-synapses pruning results in a modification function that has a similar form to the NR-driven modification function studied in the previous Section: both strategies remove the weakest synapses and linearly modify the remaining synapses in a similar manner. In the case of *limited overall synaptic strength* ($k > 0$ in Eq. 6), the optimal $g$ satisfies

$$z - 2\gamma_1 [g(z) - E(g(z))] - \gamma_2 kg(z)^{k-1} = 0 \quad , \quad (8)$$

and thus for $k = 1$ and $k = 2$ the optimal modification function is again linear. For $k > 2$ a sublinear modification function is optimal, where $g$ is a function of $z^{1/(k-1)}$,

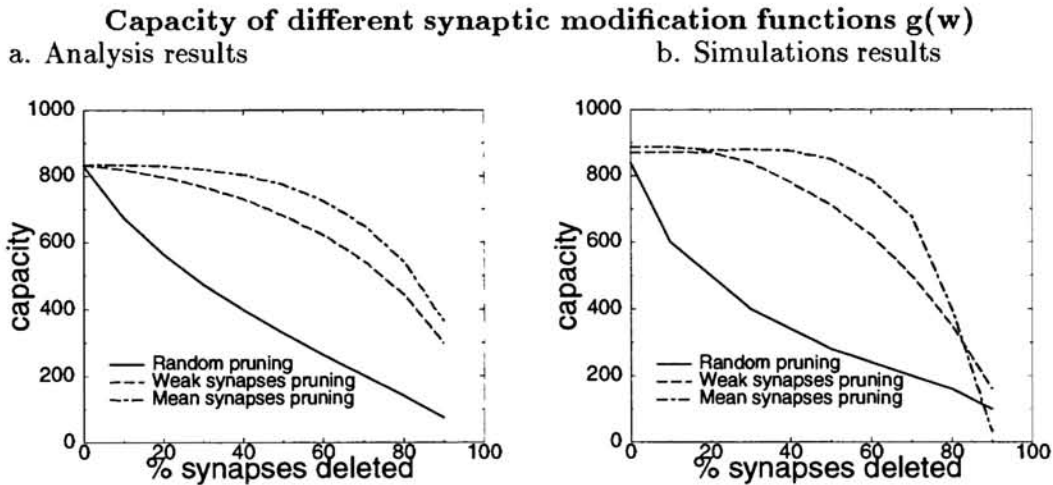

Figure 3: Comparison between performance of different modification strategies as a function of the deletion level (percentage of synapses pruned). Capacity is measured as the number of patterns that can be stored in the network ($N = 2000$) and be recalled almost correctly ($m > 0.95$) from a degraded pattern ($m_0 = 0.80$).

and is thus unbounded for all $k$. Therefore, in our model, bounds on the synaptic efficacies are not dictated by the optimization process. Their computational advantage arises from their effect on preserving memory capacity in face of ongoing synaptic pruning.

## 5   Discussion

By studying *NR-driven synaptic modification* in the framework of associative memory networks, we show that NR prunes the weaker synapses and modifies the remaining synapses in a sigmoidal manner. The critical variables that govern the pruning process are the *degradation dimension* and the *upper synaptic bound*. Our results show that **in the correct range of these parameters, NR implements a near optimal strategy, maximizing memory capacity in the sparse connectivity levels observed in the brain**.

A fundamental requirement of central nervous system development is that the system should continuously function, while undergoing major structural and functional developmental changes. It has been proposed that a major functional role of neuronal down-regulation during early infancy is to maintain neuronal activity at its baseline levels while facing continuous increase in the number and efficacy of synapses [3]. Focusing on up-regulation, our work shows that NR has another important interesting effect: that of modifying and pruning synapses in a continuously optimal manner. Neuronally regulated synaptic modifications may play the same role also in the peripheral nervous system: It was recently shown that in the neuro-muscular junction the muscle regulates its incoming synapses in a way similar to NR [9]. Our analysis suggests this process may be the underlying cause for the finding that synapses in the neuro-muscular junction are either strengthened or pruned according to their initial efficacy [10].

The significance of our work goes beyond understanding synaptic organization and remodeling in the associative memory models studied in this paper. Our analysis bears relevance to two other fundamental paradigms: Hetero Associative memory and self organizing maps, sharing the same basic synaptic structure of storing as-

sociations between sets of patterns via a Hebbian learning rule.

Combining the investigation of a biologically identified mechanism with the analytic study of performance optimization in neural network models, this paper shows the biologically plausible and beneficial role of weight dependent synaptic pruning. Thus, in addition to the known effects of Hebbian learning, neuronal regulation may play an important role in the self-organization of brain networks during development.

## Footnotes

[1] As the weights are normally distributed with expectation $Ma > 0$ and standard deviation $O(\sqrt{M})$, the probability of a negative synapse vanishes as $M$ goes to infinity (and is negligible already for several dozens of memories in the parameters' range used here).

# References

[1] G.M. Innocenti. Exuberant development of connections and its possible permissive role in cortical evolution. *Trends Neurosci*, 18:397–402, 1995.

[2] G. Chechik, I. Meilijson, and E. Ruppin. Synaptic pruning during development: A computational account. *Neural Computation. In press.*, 1998.

[3] G.G. Turrigano, K. Leslie, N. Desai, and S.B. Nelson. Activity dependent scaling of quantal amplitude in neocoritcal pyramidal neurons. *Nature*, 391(6670):892–896, 1998.

[4] D. Horn, N. Levy, and E. Ruppin. Synaptic maintenance via neuronal regulation. *Neural Computation*, 10(1):1–18, 1998.

[5] J.R. Wolff, R. Laskawi, W.B. Spatz, and M. Missler. Structural dynamics of synapses and synaptic components. *Behavioral Brain Research*, 66(1-2):13–20, 1995.

[6] M.V. Tsodyks and M. Feigel'man. Enhanced storage capacity in neural networks with low activity level. *Europhys. Lett.*, 6:101–105, 1988.

[7] Per E. Roland. *Brain Activation*. Willey-Liss, 1993.

[8] I. Meilijson and E. Ruppin. Optimal firing in sparsely-connected low-activity attractor networks. *Biological cybernetics*, 74:479–485, 1996.

[9] G.W. Davis and C.S. Goodman. Synapse-specific control of synaptic efficacy at the terminals of a single neuron. *Nature*, 392(6671):82–86, 1998.

[10] H. Colman, J. Nabekura, and J. W. Lichtman. Alterations in synaptic strength preceding axon withdrawal. *Science*, 275(5298):356–361, 1997.
